# Support Vector Machines on a Budget

**Ofer Dekel** and **Yoram Singer**
School of Computer Science and Engineering
The Hebrew University
Jerusalem 91904, Israel
{oferd,singer}@cs.huji.ac.il

## Abstract

The standard Support Vector Machine formulation does not provide its user with the ability to explicitly control the number of support vectors used to define the generated classifier. We present a modified version of SVM that allows the user to set a budget parameter $B$ and focuses on minimizing the loss attained by the $B$ worst-classified examples while ignoring the remaining examples. This idea can be used to derive sparse versions of both $L1$-SVM and $L2$-SVM. Technically, we obtain these new SVM variants by replacing the 1-norm in the standard SVM formulation with various interpolation-norms. We also adapt the SMO optimization algorithm to our setting and report on some preliminary experimental results.

## 1  Introduction

The $L1$ *Support Vector Machine* ($L1$-SVM or SVM for short) [1, 2, 3] is a powerful technique for learning binary classifiers from examples. Given a training set $\{(\mathbf{x}_i, y_i)\}_{i=1}^m$ and a positive semi-definite kernel $K$, the SVM solution is a hypothesis of the form $h(\mathbf{x}) = \text{sign}\left(\sum_{i \in S} \alpha_i y_i K(\mathbf{x}_i, \mathbf{x}) + b\right)$, where $S$ is a subset of $\{1, \ldots, m\}$, $\{\alpha_i\}_{i \in S}$ are real valued weights, and $b$ is a bias term. The set $S$ defines the *support* of the classifier, namely, the set of examples that actively participate in the classifier's definition. The examples in this set are called *support vectors*, and we say that the SVM solution is sparse if the fraction of support vectors ($|S|/m$) is reasonably small.

Our first concern is usually with the accuracy of the classifier. However, in some applications, the size of the support is equally important. Assuming that the kernel operator $K$ can be evaluated in constant time, the time-complexity of evaluating the classifier on a new instance is linear in the size of $S$. Therefore, a large support defines a slow classifier. Classification speed is often important and plays an especially critical role in real-time systems. For example, a classifier that drives a phoneme detector in a speech recognition system is evaluated hundreds of times a second. If this classifier does not manage to keep up with the rate at which the speech signal is acquired then its classifications are useless, regardless of their accuracy. The size of the support also naturally determines the amount of memory required to store the classifier. If a classifier is intended to run in a device with a limited memory, such as a mobile telephone, there may be a physical limit on the amount of memory available to store support vectors. The size of $S$ may also effect the time required to train an SVM classifier. Most modern SVM learning algorithms are *active set methods*, namely, on every step of the training process, only a small set of active training examples are taken into account. Knowing the size of $S$ ahead of time would enable us to optimize the size of the active set and possibly gain a significant speed-up in the training process.

The SVM mechanism does not give us explicit control over the size of the support. The user-defined parameters of SVM have some influence on the size of $S$, but we often require more than this. Specifically, we would like the ability to specify a *budget parameter*, $B$, which directly controls the number of support vectors used to define the SVM solution. In this paper, we address this issue

and present *budget-SVM*, a minor modification to the standard $L1$-SVM formulation that allows the user to set a budget parameter. The budget-SVM optimization problem focuses only on the $B$ worst-classified examples in the training set, ignoring all other examples.

The problem of sparsity becomes even more critical when it comes to $L2$-SVM [3], a variant of the SVM problem that tends to have dense solutions. $L2$-SVM is sometimes preferred over $L1$-SVM because it exhibits good generalization properties, as well as other desirable statistical characteristics [4]. We derive the budget-$L2$-SVM formulation by following the same technique used to derive budget-$L1$-SVM.

The technique used to derive these SVM variants is as follows. We begin by generalizing the $L1$-SVM formulation by replacing the 1-norm with an arbitrary norm. We obtain a general framework for SVM-type problems, which we nickname *Any-Norm-SVM*. Next, we turn to the $K$-method of norm interpolation to obtain the $1 - \infty$ interpolation-norm and the $2 - \infty$ interpolation-norm, and use these norms in the Any-Norm-SVM framework. These norms have the property that they depend only on the absolutely-largest elements of the vector. We rely on this property and show that our SVM variants construct sparse solutions. For each of these norms, we present a simple modification of the SMO algorithm [5], which efficiently solves the respective optimization problem.

**Related Work**   The problem of approximating the SVM solution using a reduced set of examples has received much previous attention [6, 7, 8, 9]. This technique takes a two-step approach: begin by training a standard SVM classifier, perhaps obtaining a dense solution. Then, try to find a sparse classifier which minimizes the $L2$ distance to the SVM solution. A potential drawback of this approach is that once the SVM solution has been found, the distribution from which the training set was sampled no longer plays a role in the learning process. This ignores the fact that shifting the SVM classifier by a fixed amount in different directions may have dramatically different consequences on classification accuracy. We overcome this problem by taking the approach of [10] and reformulating the SVM optimization problem itself in a way that promotes sparsity. Another technique used to obtain a sparse kernel-machine takes advantage of the inherent sparsity of linear programming solutions, and formalizes the kernel-machine learning problem as a linear program [11]. This approach, often called *LP-SVM* or *Sparse-SVM*, has been shown to generally construct sparse solutions, but still lacks the ability to introduce an explicit budget parameter. Yet another approach involves randomly selecting a subset of the training set to serve as support vectors [12]. The problem of learning a kernel-machine on a budget also appears in the online-learning mistake-bound framework, and it is there where the term "learning on a budget" was coined  [13]. Two recent papers [14, 15] propose online kernel-methods on a budget with an accompanying theoretical mistake-bound.

This paper is organized as follows. We present the generalized Any-Norm-SVM framework in Sec. 2. We discuss the $K$-method of norm interpolation in Sec. 3 and put various interpolation norms to use within the Any-Norm-SVM framework in Sec. 4. Then, in Sec. 5, we present some preliminary experiments that demonstrate how the theoretical properties of our approach translate into practice. We conclude with a discussion in Sec. 6. Due to the lack of space, some of the proofs are omitted from this paper.

## 2   Any-Norm SVM

Let $\{(\mathbf{x}_i, y_i)\}_{i=1}^m$ be a training set, where every $\mathbf{x}_i$ belongs to an instance space $\mathcal{X}$ and every $y_i \in \{-1, +1\}$. Let $K : \mathcal{X} \times \mathcal{X} \to \mathbb{R}$ be a positive semi-definite kernel, and let $\mathcal{H}$ be its corresponding Reproducing Kernel Hilbert Space (RKHS) [16], with inner product $\langle \cdot, \cdot \rangle_{\mathcal{H}}$. The $L1$ Support Vector Machine is defined as the solution to the following convex optimization problem:

$$\min_{f \in \mathcal{H},\, b \in \mathbb{R},\, \boldsymbol{\xi} \geq 0} \tfrac{1}{2} \langle f, f \rangle_{\mathcal{H}} + C \|\boldsymbol{\xi}\|_1 \quad \text{s.t.} \quad \forall\, 1 \leq i \leq m \;\; y_i\big(f(\mathbf{x}_i) + b\big) \geq 1 - \xi_i \;, \qquad (1)$$

where $\boldsymbol{\xi}$ is a vector of $m$ slack variables, and $C$ is a positive constant that controls the tradeoff between the complexity of the learned classifier and how well it fits the training data. The value of $\xi_i$ is sometimes referred to as the *hinge-loss* attained by the SVM classifier on example $i$. The 1-norm, defined by $\|\boldsymbol{\xi}\|_1 = \sum_{i=1}^m |\xi_i|$, is used to combine the individual hinge-loss values into a single number.

$L2$-SVM is a variant of the optimization problem defined above, defined as follows:

$$\min_{f \in \mathcal{H}, b \in \mathbb{R}, \boldsymbol{\xi} \geq 0} \tfrac{1}{2} \langle f, f \rangle_{\mathcal{H}} + C \|\boldsymbol{\xi}\|_2^2 \quad \text{s.t.} \quad \forall \, 1 \leq i \leq m \quad y_i \big( f(\mathbf{x}_i) + b \big) \geq 1 - \xi_i \ .$$

This formulation differs from the $L1$ formulation in that the 1-norm is replaced by the *squared* 2-norm, defined by $\|\boldsymbol{\xi}\|_2^2 = \sum_{i=1}^m \xi_i^2$. In this section, we take this idea even farther, and allow the 1-norm of $L1$-SVM to be replaced by any norm. Formally, let $\| \cdot \|$ be an arbitrary norm defined on $\mathbb{R}^m$. Recall that a norm is a real valued operator such that for every $\mathbf{v} \in \mathbb{R}^m$ and $\lambda \in \mathbb{R}$ it holds that $\|\lambda \mathbf{v}\| = |\lambda| \|\mathbf{v}\|$ (positive homogeneity), $\|\mathbf{v}\| \geq 0$ and $\|\mathbf{v}\| = 0$ if and only if $\mathbf{v} = 0$ (positive definiteness), and that satisfies the triangle inequality. Now consider the following optimization problem:

$$\min_{f \in \mathcal{H}, b \in \mathbb{R}, \boldsymbol{\xi} \geq 0} \tfrac{1}{2} \langle f, f \rangle_{\mathcal{H}} + C \|\boldsymbol{\xi}\| \quad \text{s.t.} \quad \forall \, 1 \leq i \leq m \quad y_i \big( f(\mathbf{x}_i) + b \big) \geq 1 - \xi_i \ . \tag{2}$$

$L1$-SVM is recovered by setting $\| \cdot \|$ to be the 1-norm. Setting $\| \cdot \|$ to be the 2-norm induces an optimization problem which is close in nature to $L2$-SVM, but not identical to it since the 2-norm is not squared. Combining the positive homogeneity property of $\| \cdot \|$ with the fact that it satisfies the triangle inequality ensures that the objective function of Eq. (2) is convex.

An important class of norms used extensively in our derivation is the family of $p$-norms, defined for every $p \geq 1$ by $\|\mathbf{v}\|_p = (\sum_{j=1}^m |v_j|^p)^{1/p}$. A special member of this family is the $\infty$-norm, which is defined by $\|\mathbf{v}\|_\infty = \lim_{p \to \infty} \|\mathbf{v}\|_p$ and can be shown to be equivalent to $\max_j |v_j|$. We also use the notion of norm duality. Every norm on $\mathbb{R}^m$ has a dual norm which is also defined on $\mathbb{R}^m$. The dual norm of $\| \cdot \|$ is denoted by $\| \cdot \|^\star$ and given by

$$\|\mathbf{u}\|^\star \;=\; \max_{\mathbf{v} \in \mathbb{R}^m} \frac{\mathbf{u} \cdot \mathbf{v}}{\|\mathbf{v}\|} \;=\; \max_{\mathbf{v} \in \mathbb{R}^m \, : \, \|\mathbf{v}\| = 1} \mathbf{u} \cdot \mathbf{v} \ . \tag{3}$$

As its name implies, $\| \cdot \|^\star$ also satisfies the requirements of a norm. For example, Hölder's inequality [17] states that the dual of $\| \cdot \|_p$ is the norm $\| \cdot \|_q$, where $q = p/(p-1)$. The dual of the 1-norm is the $\infty$-norm and vice versa.

Using the definition of the dual norm, we now state the dual optimization problem of Eq. (2):

$$\max_{\boldsymbol{\alpha} \geq 0} \sum_{i=1}^m \alpha_i - \tfrac{1}{2} \sum_{i=1}^m \sum_{j=1}^m \alpha_i \alpha_j y_i y_j K(\mathbf{x}_i, \mathbf{x}_j) \quad \text{s.t.} \quad \sum_{i=1}^m y_i \alpha_i = 0 \ \text{ and } \ \|\alpha\|^\star \leq C \ . \tag{4}$$

As a first sanity check, note that if $\| \cdot \|$ in Eq. (2) is chosen to be the 1-norm, then $\| \cdot \|^\star$ is the $\infty$-norm, and the constraint $\|\alpha\|^\star \leq C$ reduces to the familiar box-constraint of $L1$-SVM [3]. The proof that Eq. (2) and Eq. (4) are indeed dual optimization problems relies on basic techniques in convex analysis [18], and is omitted due to the lack of space. Moreover, it can be shown that the solution to Eq. (2) takes the form $f(\cdot) = \sum_{i=1}^m \alpha_i y_i K(\mathbf{x}_i, \cdot)$, and that strong duality holds regardless of the norm used. This allows us to forget about the primal problem in Eq. (2) and to focus on solving the dual problem in Eq. (4). As with $L1$-SVM, the bias term, $b$, cannot be directly extracted from the solution of the dual. The standard techniques used to find $b$ in $L1$-SVM apply here as well [3].

We note that the Any-Norm-SVM formulation is not fundamentally different from the original $L1$-SVM formulation. Both optimization problems have convex objective functions and linear constraints. More importantly, the only difference between their respective duals is in the dual-norm constraint. Specifically, the objective function in Eq. (4) is a concave quadratic function for any choice of $\| \cdot \|$. These facts enable us to efficiently solve the problem in Eq. (4) for any kernel $K$ and any norm using techniques similar to those used to solve the standard $L1$-SVM problem.

## 3   Interpolation Norms

In the previous section, we acquired the ability to replace the 1-norm in the definition of $L1$-SVM with an arbitrary norm. We now use Peetre's $K$-method of norm interpolation [19] to obtain norms that promote the sparsity of the generated classifier. The $K$-method is a technique for smoothly interpolating between a pair of norms. Let $\| \cdot \|_{p_1} : \mathbb{R}^m \to \mathbb{R}_+$ and $\| \cdot \|_{p_2} : \mathbb{R}^m \to \mathbb{R}_+$ be two $p$-norms, and let $\| \cdot \|_{q_1}$ and $\| \cdot \|_{q_2}$ be their respective duals. *Peetre's $K$-functional* with respect to $p_1$ and $p_2$, and with respect to the constant $t > 0$, is defined to be

$$\|\mathbf{v}\|_{K(p_1, p_2, t)} \;=\; \min_{\mathbf{w}, \mathbf{z} \, : \, \mathbf{w} + \mathbf{z} = \mathbf{v}} \Big( \|\mathbf{w}\|_{p_1} + t \|\mathbf{z}\|_{p_2} \Big) \ . \tag{5}$$

*Peetre's J-functional* with respect to $q_1$, $q_2$, and with respect to the constant $s > 0$, is given by

$$\|\mathbf{u}\|_{J(q_1,q_2,s)} \;=\; \max\left\{\|\mathbf{u}\|_{q_1},\; s\,\|\mathbf{u}\|_{q_2}\right\}\;. \tag{6}$$

The $J$-functional is obviously a norm: the properties of a norm all follow immediately from the fact that $\|\cdot\|_{q_1}$ and $\|\cdot\|_{q_2}$ posses these properties. $\|\cdot\|_{K(p_1,p_2,t)}$ is also a norm, and moreover, $\|\cdot\|_{K(p_1,p_2,t)}$ and $\|\cdot\|_{J(q_1,q_2,s)}$ are dual to each other when $t = 1/s$. This fact can be proven using elementary calculus, and this proof is omitted due to the lack of space.

We use the $K$-method to interpolate between the 1-norm and the $\infty$-norm, and to interpolate between the 2-norm and the $\infty$-norm. To gain some intuition on the behavior of these interpolation-norms, first note that for any $p \geq 1$ and any $\mathbf{v} \in \mathbb{R}^m$ it holds that $\max_i |v_i|^p \leq \sum_{i=1}^m |v_i|^p \leq m \max_i |v_i|^p$, and therefore $\|\mathbf{v}\|_\infty \leq \|\mathbf{v}\|_p \leq m^{1/p}\|\mathbf{v}\|_\infty$. An immediate consequence of this is that $\|\cdot\|_{K(p,\infty,t)} \equiv \|\cdot\|_\infty$ when $0 < t \leq 1$ and that $\|\cdot\|_{K(p,\infty,t)} \equiv \|\cdot\|_p$ when $m^{1/p} \leq t$. In other words, the interesting range of $t$ for the $1 - \infty$ interpolation-norm is $[1, m]$, and for the $2 - \infty$ interpolation-norm is $[1, \sqrt{m}]$.

Next, we prove a theorem which states that interpolating a $p$-norm with the $\infty$-norm is approximately equivalent to restricting that $p$-norm to the absolutely-largest components of the vector. Specifically, the $1 - \infty$ interpolation norm with parameter $t$ (with $t$ chosen to be an integer in $[1, m]$) is precisely equivalent to taking the sum of the absolute values of the $t$ absolutely-greatest elements of the vector.

**Theorem 1.** *Let $\mathbf{v}$ be an arbitrary vector in $\mathbb{R}^m$ and let $\pi$ be a permutation on $\{1, \ldots, m\}$ such that $|v_{\pi(1)}| \geq \ldots \geq |v_{\pi(m)}|$. Then for any integer $B$ in $\{1, \ldots, m\}$ it holds that $\|\mathbf{v}\|_{K(1,\infty,B)} = \sum_{i=1}^B |v_{\pi(i)}|$, and for any $1 \leq p < \infty$, if $t = B^{1/p}$ then it holds that*

$$\left(\textstyle\sum_{i=1}^B |v_{\pi(i)}|^p\right)^{1/p} \;\leq\; \|\mathbf{v}\|_{K(p,\infty,t)} \;\leq\; \left(\textstyle\sum_{i=1}^B |v_{\pi(i)}|^p\right)^{1/p} + B^{1/p}|v_{\pi(B)}|\;.$$

*Proof.* Beginning with the lower bound, let $\mathbf{w}$ and $\mathbf{z}$ be such that $\mathbf{w} + \mathbf{z} = \mathbf{v}$. Then

$$
\begin{aligned}
\left(\textstyle\sum_{i=1}^B |v_{\pi(i)}|^p\right)^{1/p} &= \left(\textstyle\sum_{i=1}^B |w_{\pi(i)} + z_{\pi(i)}|^p\right)^{1/p} \\
&\leq \left(\textstyle\sum_{i=1}^B |w_{\pi(i)}|^p\right)^{1/p} + \left(\textstyle\sum_{i=1}^B |z_{\pi(i)}|^p\right)^{1/p} \\
&\leq \left(\textstyle\sum_{i=1}^B |w_{\pi(i)}|^p\right)^{1/p} + \left(B \max_i |z_i|^p\right)^{1/p} \\
&\leq \left(\textstyle\sum_{i=1}^m |w_i|^p\right)^{1/p} + t\|\mathbf{z}\|_\infty\;,
\end{aligned}
$$

where the first inequality is the triangle inequality for the $p$-norm. Since the above holds for any $\mathbf{w}$ and $\mathbf{z}$ such that $\mathbf{w} + \mathbf{z} = \mathbf{v}$, it also holds for the pair which minimizes $(\sum_{i=1}^m |w_i|^p)^{1/p} + t\|\mathbf{z}\|_\infty$, and which defines $\|\mathbf{v}\|_{K(p,\infty,t)}$. Therefore, we have that,

$$\left(\textstyle\sum_{i=1}^B |v_{\pi(i)}|^p\right)^{1/p} \;\leq\; \|\mathbf{v}\|_{K(p,\infty,t)}\;. \tag{7}$$

Turning to the upper bound, let $\phi = |v_{\pi(B)}|$, and define for all $1 \leq i \leq m$, $\bar{w}_i = \text{sign}(v_i)\max\{0, |v_i| - \phi\}$ and $\bar{z}_i = \text{sign}(v_i)\min\{|v_i|, \phi\}$. Note that $\bar{\mathbf{w}} + \bar{\mathbf{z}} = \mathbf{v}$, and that

$$\sum_{i=1}^B |v_{\pi(i)}| \;=\; \|\bar{\mathbf{w}}\|_1 + B\|\bar{\mathbf{z}}\|_\infty\;.$$

This proves that $\|\mathbf{v}\|_{K(1,\infty,B)} \leq \sum_{i=1}^B |v_{\pi(i)}|$ and together with Eq. (7) we have proven our claim for $p = 1$. Moving on to the case of an arbitrary $p$, we have that

$$\|\mathbf{v}\|_{K(p,\infty,t)} \;=\; \min_{\mathbf{w}+\mathbf{z}=\mathbf{v}}\left(\|\mathbf{w}\|_p + t\|\mathbf{z}\|_\infty\right) \;\leq\; \|\bar{\mathbf{w}}\|_p + t\|\bar{\mathbf{z}}\|_\infty\;.$$

Since the absolute value of each element in $\bar{\mathbf{w}}$ is at most as large as the absolute value of the corresponding element of $\mathbf{v}$, and since $\bar{w}_{\pi(r+1)} = \ldots = \bar{w}_{\pi(m)} = 0$, we have that $\|\bar{\mathbf{w}}\|_p \leq (\sum_{i=1}^B |v_{\pi(i)}|^p)^{1/p}$. By definition, $\|\bar{\mathbf{z}}\|_\infty = \phi = |v_{\pi(B)}|$. This proves that $\|\mathbf{v}\|_{K(p,\infty,t)} \leq (\sum_{i=1}^B |v_{\pi(i)}|^p)^{1/p} + t|v_{\pi(B)}|$ and together with Eq. (7) this concludes our proof for arbitrary $p$. $\square$

# 4 Deriving Concrete Algorithms from the General Framework

Our first concrete algorithm is budget-$L1$-SVM, obtained by plugging the $1-\infty$ interpolation-norm with parameter $B$ into the general Any-Norm-SVM framework. Relying on Thm. 1, we know that this norm takes into account only the $B$ largest values in $\boldsymbol{\xi}$. Since $\boldsymbol{\xi}$ measures how badly each example is misclassified, the budget-$L1$-SVM problem essentially optimizes the soft-margin with respect to the $B$ worst-classified examples. We now show that this property promotes the sparsity of the budget-$L1$-SVM solution.

If there are less than $B$ examples for which $y_i(f(\mathbf{x}_i)+b) < 1$, then the KKT conditions of optimality immediately imply that the number of support vectors is less than $B$. This holds true for every instance of the Any-Norm-SVM framework, and is proven for $L1$-SVM in [3]. Therefore, we focus on the more interesting case, where $y_i(f(\mathbf{x}_i) + b) < 1$ for *at least $B$ examples*.

**Theorem 2.** *Let $B$ be an integer in $\{1, \ldots, m\}$. Let $(f, b, \boldsymbol{\xi}, \boldsymbol{\alpha})$ be an optimal primal-dual solution of the primal problem in Eq. (2) and the dual problem in Eq. (4), where $\|\cdot\|$ is chosen to be the $1-\infty$ interpolation-norm with parameter $B$. Define $\mu_i = y_i(f(\mathbf{x}_i) + b)$ and let $\pi$ be a permutation of $\{1, \ldots, m\}$ such that $\mu_{\pi(1)} \leq \ldots \leq \mu_{\pi(m)}$. Assume that $\mu_{\pi(B)} < 1$. Then, $\alpha_k = 0$ if $\mu_{\pi(B)} < \mu_k$.*

*Proof.* We begin the proof by redefining $\xi_i = \max\{1 - \mu_i, 0\}$ for all $1 \leq i \leq m$ and noting that $(f, b, \boldsymbol{\xi}, \boldsymbol{\alpha})$ remains a primal-dual solution to our problem. The benefit of starting with this specific solution is that $\xi_{\pi(1)} \geq \ldots \geq \xi_{\pi(m)}$. Let $k$ be an index such that $\mu_{\pi(B)} < \mu_k$ and define $\xi'_k = \frac{1}{2}(\xi_k + \xi_{\pi(B)})$. Moreover, let $\boldsymbol{\xi}'$ be the vector obtained by replacing the $k$'th coordinate in $\boldsymbol{\xi}$ with $\xi'_k$, or in other words, $\boldsymbol{\xi}' = (\xi_1, \ldots, \xi'_k, \ldots, \xi_m)$. Using the assumption that $\mu_{\pi(B)} < 1$, we know that $\xi_{\pi(B)} > 0$, and since $\mu_k > \mu_{\pi(B)}$ we get that $\xi_k < \xi_{\pi(B)}$. We can now draw two conclusions. First, $\xi_{\pi(1)} \geq \ldots \geq \xi_{\pi(B)} > \xi'_k$ and therefore $\|\boldsymbol{\xi}'\|_{K(1,\infty,B)} = \|\boldsymbol{\xi}\|_{K(1,\infty,B)}$. Second, $\xi_k < \xi'_k$ and therefore $\boldsymbol{\xi}'$ satisfies the constraints of Eq. (2). Overall, we obtain that $(f, b, \boldsymbol{\xi}', \boldsymbol{\alpha})$ is also a primal-dual solution to our problem. Moreover, we know that $1 - \mu_k < \xi'_k$. Using the KKT complementary slackness condition, it follows that $\alpha_k$, the Lagrange multiplier corresponding to this constraint, must equal 0. $\qquad\square$

Defining $\mu_i$ and $\pi$ as above, a simple corollary of Thm. 2 is that the number of support vectors is upper bounded by $B$ in the case that $\mu_{\pi(B)} \neq \mu_{\pi(B+1)}$.

From our discussion in Sec. 3, we know that the dual of the $1 - \infty$ interpolation-norm is the function $\max\{\|\mathbf{u}\|_\infty, (1/B)\|\mathbf{u}\|_1\}$. Plugging this definition into Eq. (4) gives us the dual optimization problem of budget-$L1$-SVM. The constraint $\|\boldsymbol{\alpha}\|^\star \leq C$ simplifies to $\alpha_i \leq C$ for all $i$ and $\sum_{i=1}^m \alpha_i \leq BC$. To numerically solve this optimization problem, we turn to the *Sequential Minimal Optimization* (SMO) [5] technique. We briefly describe the SMO technique, and then discuss its adaptation to our setting. SMO is an iterative process, which on every iteration selects a pair of dual variables, $\alpha_k$ and $\alpha_l$, and optimizes the dual problem with respect to them, leaving all other variables fixed. The choice of the two variables is determined by a heuristic [5], and their optimal values are calculated analytically. Assume that we start with a vector $\boldsymbol{\alpha}$ which is a feasible point of the optimization problem in Eq. (4). When restricted to the two active variables, $\alpha_k$ and $\alpha_l$, the constraint $\sum_{i \neq k,l} \alpha_i y_i = 0$ simplifies to $\alpha_k^{\text{new}} y_k + \alpha_l^{\text{new}} y_l = \alpha_k^{\text{old}} y_k + \alpha_l^{\text{old}} y_l$. Put another way, we can slightly overload our notation and define the linear functions

$$\alpha_k(\lambda) = \alpha_k + \lambda y_k \quad \text{and} \quad \alpha_l(\lambda) = \alpha_l - \lambda y_l \ , \tag{8}$$

and find the single variable $\lambda$ which maximizes our constrained optimization problem. Since the constraints in Eq. (4) define a convex and bounded feasible set, the intersection of the linear equalities in Eq. (8) with this feasible set restricts $\lambda$ to an interval. The objective function, as a function of the single variable $\lambda$, takes the form $\mathcal{O}(\lambda) = P\lambda^2 + Q\lambda + c$, where $c$ is a constant,

$$P \ = \ K(\mathbf{x}_k, \mathbf{x}_l) - \tfrac{1}{2}K(\mathbf{x}_k, \mathbf{x}_k) - \tfrac{1}{2}K(\mathbf{x}_l, \mathbf{x}_l) \ , \quad Q \ = \ \big(y_k - f(\mathbf{x}_k)\big) - \big(y_l - f(\mathbf{x}_l)\big) \ ,$$

and $f$ is the current function in the RKHS ($f \equiv \sum_{i=1}^m \alpha_i y_i K(\mathbf{x}_i, \cdot)$). Maximizing the objective function in Eq. (4) with respect to $\alpha_k$ and $\alpha_l$ is equivalent to maximizing $\mathcal{O}(\lambda)$ with respect to $\lambda$ over an interval. $P$ equals minus the Euclidean distance between the functions $K(\mathbf{x}_k, \cdot)$ and

$K(\mathbf{x}_l, \cdot)$ in the RKHS, and is therefore a negative number. Therefore, $\mathcal{O}(\lambda)$ is a concave function which attains a single (unconstrained) maximum. This maximum can be found analytically by

$$0 \;=\; \frac{\partial \mathcal{O}(\lambda)}{\partial \lambda} \;=\; 2P\lambda + Q \quad\Rightarrow\quad \lambda \;=\; \frac{-Q}{2P} \;. \tag{9}$$

If this unconstrained optimum falls inside the feasible interval, then it is equivalent to the constrained optimum. Otherwise, the constrained optimum falls on one of the two end-points of the interval. Thus, we are left with the task of finding these end-points. To do so, we consider the remaining constraints:

$$\text{(I)} \quad \begin{array}{l} \alpha_k(\lambda) \geq 0 \\ \alpha_l(\lambda) \geq 0 \end{array} \qquad \text{(II)} \quad \begin{array}{l} \alpha_k(\lambda) \leq C \\ \alpha_l(\lambda) \leq C \end{array} \qquad \text{(III)} \; \alpha_k(\lambda) + \alpha_l(\lambda) \;\leq\; BC - \sum_{i \neq k,l} \alpha_i \;.$$

The constraints in (I) translate to

$$\begin{array}{llll} y_k = -1 & \Rightarrow & \lambda \leq \alpha_k & \qquad y_k = +1 \;\; \Rightarrow \;\; \lambda \geq -\alpha_k \\ y_l = -1 & \Rightarrow & \lambda \geq -\alpha_l & \qquad y_l = +1 \;\; \Rightarrow \;\; \lambda \leq \alpha_l \;. \end{array} \tag{10}$$

The constraints in (II) translate to

$$\begin{array}{llll} y_k = -1 & \Rightarrow & \lambda \geq \alpha_k - C & \qquad y_k = +1 \;\; \Rightarrow \;\; \lambda \leq C - \alpha_k \\ y_l = -1 & \Rightarrow & \lambda \leq C - \alpha_l & \qquad y_l = +1 \;\; \Rightarrow \;\; \lambda \geq \alpha_l - C \;. \end{array} \tag{11}$$

Constraint (III) translates to

$$\begin{array}{lll} y_k = -1 \,\wedge\, y_l = +1 & \Rightarrow & \lambda \geq \tfrac{1}{2}\left(\sum_{i=1}^{m} \alpha_i - BC\right) \\ y_k = +1 \,\wedge\, y_l = -1 & \Rightarrow & \lambda \leq \tfrac{1}{2}\left(BC - \sum_{i=1}^{m} \alpha_i\right) \;. \end{array} \tag{12}$$

Finding the end-points of the interval that confines $\lambda$ amounts to finding the smallest upper bound and the greatest lower bound in Eqs. (10,11,12). This concludes the analytic derivation of the SMO update for budget-$L1$-SVM.

**L2-SVM on a budget**    Next, we use the $2 - \infty$ interpolation-norm with parameter $t = \sqrt{B}$ in the Any-Norm-SVM framework, and obtain the budget-$L2$-SVM problem. Thm. 1 hints that setting $t = \sqrt{B}$ makes the $2 - \infty$ interpolation-norm almost equivalent to restricting the 2-norm to the top $B$ elements in the vector $\boldsymbol{\xi}$. The support size of the budget-$L2$-SVM solution is strongly correlated with the parameter $B$ although the exact relation between the two is not as clear as before. Again we begin with the dual formulation defined in Eq. (4), where the constraint $\|\boldsymbol{\alpha}\|^\star \leq C$ becomes $\max\{\|\boldsymbol{\xi}\|_2, (1/\sqrt{B})\|\boldsymbol{\xi}\|_1\} \leq C$. The intersection of this constraint with the other constraints defines a convex and bounded feasible set, and its intersection with the linear equalities in Eq. (8) defines an interval. The objective function in Eq. (4) is the same as before, so the unconstrained maximum is once again given be Eq. (9). To obtain the constrained maximum, we must find the end-points of the interval that confines $\lambda$. The dual-norm constraint can be written more explicitly as

$$\text{(I)} \; \alpha_k(\lambda) + \alpha_l(\lambda) \;\leq\; \sqrt{B}C - \sum_{i \neq k,l} \alpha_i \qquad \text{(II)} \; \alpha_k^2(\lambda) + \alpha_l^2(\lambda) \;\leq\; C^2 - \sum_{i \neq k,l} \alpha_i^2 \;.$$

Constrain (I) is similar to the constraint we had in the budget-$L1$-SVM case, and is given in terms of $\lambda$ by replacing $B$ with $\sqrt{B}$ in Eq. (12). Constraint (II) is new, and can be written in terms of $\lambda$ as $\lambda^2 + \lambda\beta + \gamma \leq 0$, where $\beta = \alpha_k y_k - \alpha_l y_l$ and $\gamma = \tfrac{1}{2}(\sum_{i=1}^{m} \alpha_i^2 - C^2)$. It can be written even more explicitly as

$$\lambda \;\leq\; \tfrac{1}{2}\left(-\beta + \sqrt{\beta^2 - 4\gamma}\right) \quad \text{and} \quad \lambda \;\geq\; \tfrac{1}{2}\left(-\beta - \sqrt{\beta^2 - 4\gamma}\right) \;. \tag{13}$$

In addition, we still have the constraint $\boldsymbol{\alpha} \geq 0$, which is common to every instance of the Any-Norm-SVM framework. This constraint is given in terms of $\lambda$ in Eq. (10). Overall, the end-points of the interval we are searching for are found by taking the smallest upper bound and the greatest lower bound in Eqs. (10,13) and Eq. (12) with $B$ replaced by $\sqrt{B}$.

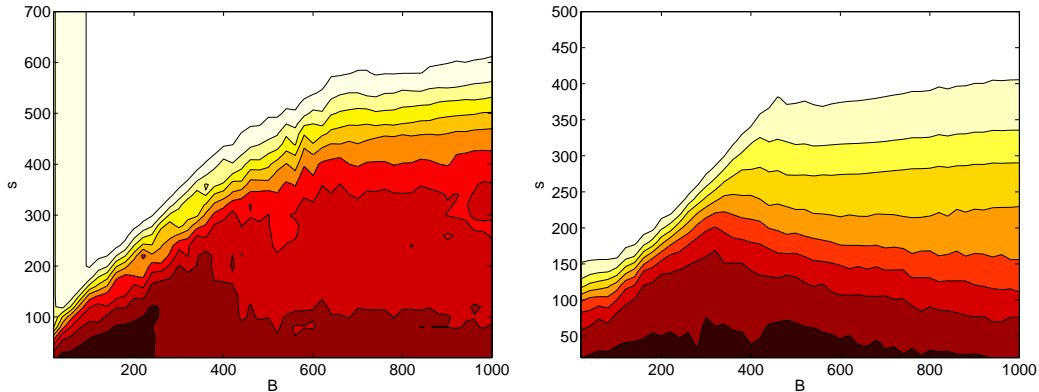

Figure 1: Average test error of budget-$L1$-SVM (left) and budget-$L2$-SVM (right) for different values of the budget parameter $B$ and the pruning parameter $s$ (all but $s$ weights in $\boldsymbol{\alpha}$ are set to zero). The test error in the darkest region is roughly 50%, and in the lightest region is roughly 5%.

## 5  Experiments

Many existing solvers for the standard $L1$-SVM problem define a positive threshold value close to zero and replace every weight that falls below this threshold with zero. This heuristic significantly reduces the time required for the algorithm to converge. In our setting, a more natural way to speed up the learning process is to run the iterative SMO optimization algorithm for a fixed number of iterations and then to keep only the $B$ largest weights, setting the $m - B$ remaining weights to zero. This pruning heuristic enforces the budget constraint in a brute-force way, and can be equally applied to any kernel-machine. However, the natural question is how much will the pruning heuristic affect the classification accuracy of the kernel-machine it is applied to. If our technique indeed lives up to its theoretical promise, we expect the pruning heuristic to have little impact on classification accuracy. On the other hand, if we train an $L1$-SVM and it so happens that the number of large weights exceeds $B$, then applying the pruning heuristic should have a dramatic negative effect on classification accuracy. The goal of our experiments is to demonstrate that this behavior indeed occurs in practice.

We conducted our experiments using the MNIST dataset, which contains handwritten digits from the 10 digit classes. We randomly generated 50 binary classification problems by first randomly partitioning the 10 classes into two equally sized sets, and then randomly choosing a training set of 1000 examples and a test set of 4000 examples. The results reported below are averaged over these 50 problems. Although MNIST is generally thought to induce easy learning problems, the method described above generates moderately difficult learning tasks.

For each binary problem, we trained both the $L1$ and the $L2$ budget SVMs with $B = 20, 40, \ldots, 1000$. Note that $\|\boldsymbol{\xi}\|_{K(1,\infty,B)}$ grows roughly linearly with $B$, and that $\|\boldsymbol{\xi}\|_{K(2,\infty,\sqrt{B})}$ grows roughly like the square root of $B$. To compensate for this, we set $C = 10/B$ in the $L1$ case and $C = 10/\sqrt{B}$ in the $L2$ case. This heuristic choice of $C$ attempts to preserve the relative weight of the regularization term with respect to the norm term in Eq. (2), across the various values of $B$. In all of our experiments, we used a Gaussian kernel with $\sigma = 1$ (after scaling the data to have an average unit norm). For each classifier trained, we pruned away all but the $s$ largest weights, with $s = 20, 40, \ldots, 1000$, and calculated the test error. The average test error for every choice of $B$ (the budget parameter in the optimization problem) and $s$ (the number of non-zero weights kept) is summarized in Fig. 1. In practice, $s$ and $B$ should be equal, however we let $s$ take different values in our experiment to illustrate the characteristics of our approach. Note that the test-error attained by $L1$-SVM (without a budget parameter) and $L2$-SVM are represented by the top-right corners of the respective plots.

As expected, classification accuracy for any value of $B$ deteriorates as $s$ becomes small. However, the accuracy attained by $L1$-SVM and $L2$-SVM can be equally attained using significantly less support vectors.

# 6  Discussion

Using the Any-Norm-SVM framework with interesting norms enabled us to introduce a budget parameter to the SVM formulation. However, the Any-Norm framework can be used for other tasks as well. For example, we can interpolate between $L1$-SVM and $L2$-SVM by using the $1-2$ interpolation-norm. This gives the user the explicit ability to balance the trade-off between the pros and cons of these two SVM variants. In [20] it is shown that there exists a constant $c$ such that,

$$c\|\mathbf{v}\|_{K(1,2,\sqrt{r})} \ \leq \ \sum_{j=1}^{r} |v_j| \ + \ \sqrt{r}\Big(\sum_{j=r+1}^{m} v_j^2\Big)^{1/2} \ \leq \ \|\mathbf{v}\|_{K(1,2,\sqrt{r})} \ .$$

These bounds give some insight into how such an interpolation would behave. Another possible norm that can be used in our framework is the Mahalanobis norm ($\|\mathbf{v}\| = (\mathbf{v}^\top M \mathbf{v})^{1/2}$, where $M$ is a positive definite matrix), which would define a loss function that takes into account pair-wise relationships between examples.

Regarding our experiments, the rule-of-thumb we used to choose the parameter $C$ is not always optimal. It seems preferable to tune $C$ individually for each $B$ using cross-validation.

We are currently exploring extensions to our SMO variant that would quickly converge to the sparse solution without the help of the pruning heuristic. We are also considering multiplicative update optimization algorithms as an alternative to SMO.

## References

[1] B. E. Boser, I. M. Guyon, and V. N. Vapnik. A training algorithm for optimal margin classifiers. In *Proc. of the Fifth Annual ACM Workshop on Computational Learning Theory*, pages 144–152, 1992.

[2] V. N. Vapnik. *Statistical Learning Theory*. Wiley, 1998.

[3] N. Cristianini and J. Shawe-Taylor. *An Introduction to Support Vector Machines*. Cambridge University Press, 2000.

[4] P. Bartlett and A. Tewari. Sparseness vs estimating conditional probabilities: Some asymptotic results. In *Proc. of the Seventeenth Annual Conference on Computational Learning Theory*, pages 564–578, 2004.

[5] J. C. Platt. Fast training of Support Vector Machines using sequential minimal optimization. In B. Schölkopf, C. Burges, and A. Smola, editors, *Advances in Kernel Methods - Support Vector Learning*. MIT Press, 1998.

[6] C.J.C. Burges. Simplified support vector decision rules. In *Proc. of the Thirteenth International Conference on Machine Learning*, pages 71–77, 1996.

[7] E. Osuna and F. Girosi. Reducing the run-time complexity of support vector machines. In B. Schölkopf, C. Burges, and A. Smola, editors, *Advances in Kernel Methods: Support Vector Learning*, pages 271–284. MIT Press, 1999.

[8] B. Schölkopf, S. Mika, C.J.C. Burges, P. Knirsch, K-R Müller, G. Rätsch, and A.J. Smola. Input space versus feature space in kernel-based methods. *IEEE Transactions on Neural Networks*, 10(5):1000–1017, September 1999.

[9] J-H. Chen and C-S. Chen. Reducing SVM classification time using multiple mirror classifiers. *IEEE transactions on systems, man and cybernetics – part B: Cybernetics*, 34(2):1173–1183, April 2004.

[10] M. Wu, B. Schölkopf, and G. Bakir. A direct method for building sparse kernel learning algorithms. *Journal of Machine Learning Research*, 7:603–624, 2006.

[11] K.P. Bennett. Combining support vector and mathematical programming methods for classification. In *Advances in kernel methods: support vector learning*, pages 307–326. MIT Press, 1999.

[12] Y. Lee and O.L. Mangasarian. RSVM: Reduced support vector machines. In *Proc. of the First SIAM International Conference on Data Mining*, 2001.

[13] K. Crammer, J. Kandola, and Y. Singer. Online classification on a budget. In *Advances in Neural Information Processing Systems 16*, 2003.

[14] O. Dekel, S. Shalev-Shwartz, and Y. Singer. The Forgetron: A kernel-based perceptron on a fixed budget. In *Advances in Neural Information Processing Systems 18*, 2005.

[15] N. Cesa-Bianchi and C. Gentile. Tracking the best hyperplane with a simple budget perceptron. In *Proc. of the Nineteenth Annual Conference on Computational Learning Theory*, 2006.

[16] N. Aronszajn. Theory of reproducing kernels. *Transactions of the American Mathematical Society*, 68(3):337–404, May 1950.

[17] R. A. Horn and C. R. Johnson. *Matrix Analysis*. Cambridge University Press, 1985.

[18] S. Boyd and L. Vandenberghe. *Convex Optimization*. Cambridge University Press, 2004.

[19] C. Bennett and R. Sharpley. *Interpolation of Operators*. Academic Press, 1998.

[20] T. Holmstedt. Interpolation of quasi-normed spaces. *Mathematica Scandinavica*, 26:177–190, 1970.
